# Characterizing neural gain control using spike-triggered covariance

**Odelia Schwartz**
Center for Neural Science
New York University
odelia@cns.nyu.edu

**E. J. Chichilnisky**
Systems Neurobiology
The Salk Institute
ej@salk.edu

**Eero P. Simoncelli**
Howard Hughes Medical Inst.
Center for Neural Science
New York University
eero.simoncelli@nyu.edu

## Abstract

Spike-triggered averaging techniques are effective for linear characterization of neural responses. But neurons exhibit important nonlinear behaviors, such as gain control, that are not captured by such analyses. We describe a spike-triggered covariance method for retrieving suppressive components of the gain control signal in a neuron. We demonstrate the method in simulation and on retinal ganglion cell data. Analysis of physiological data reveals significant suppressive axes and explains neural nonlinearities. This method should be applicable to other sensory areas and modalities.

White noise analysis has emerged as a powerful technique for characterizing response properties of spiking neurons. A sequence of stimuli are drawn randomly from an ensemble and presented in rapid succession, and one examines the subset that elicit action potentials. This "spike-triggered" stimulus ensemble can provide information about the neuron's response characteristics. In the most widely used form of this analysis, one estimates an excitatory linear kernel by computing the spike-triggered average (STA); that is, the mean stimulus that elicited a spike [e.g., 1, 2]. Under the assumption that spikes are generated by a Poisson process with instantaneous rate determined by linear projection onto a kernel followed by a static nonlinearity, the STA provides an unbiased estimate of this kernel [3]. Recently, a number of authors have developed interesting extensions of white noise analysis. Some have examined spike-triggered averages in a reduced linear subspace of input stimuli [e.g., 4]. Others have recovered excitatory subspaces, by computing the spike-triggered covariance (STC), followed by an eigenvector analysis to determine the subspace axes [e.g., 5, 6].

Sensory neurons exhibit striking nonlinear behaviors that are not explained by fundamentally linear mechanisms. For example, the response of a neuron typically saturates for large amplitude stimuli; the response to the optimal stimulus is often suppressed by the presence of a non-optimal mask [e.g., 7]; and the kernel recovered from STA analysis may change shape as a function of stimulus amplitude [e.g., 8, 9]. A variety of these nonlinear behaviors can be attributed to gain control [e.g., 8, 10, 11, 12, 13, 14], in which neural responses are suppressively modulated by a gain signal derived from the stimulus. Although the underlying mechanisms and time scales associated with such gain control are current topics of research, the basic functional properties appear to be ubiquitous, occurring throughout the nervous system.

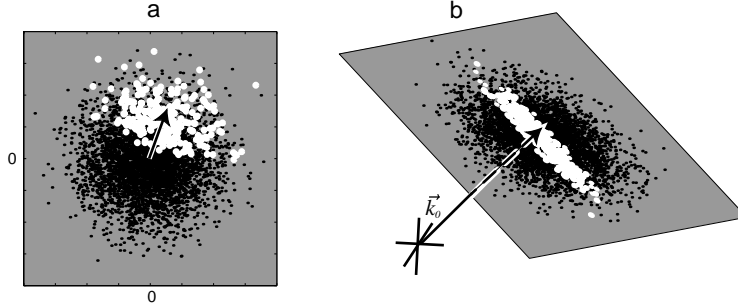

a          b

0

0

**Figure 1:** Geometric depiction of spike-triggered analyses. **a**, Spike-triggered averaging with two-dimensional stimuli. Black points indicate raw stimuli. White points indicate stimuli eliciting a spike, and the STA (black vector), which provides an estimate of $\vec{k}_0$, corresponds to their center of mass. **b**, Spike-triggered covariance analysis of suppressive axes. Shown are a set of stimuli lying on a plane perpendicular to the excitatory kernel, $\vec{k}_0$. Within the plane, stimuli eliciting a spike are concentrated in an elliptical region. The minor axis of the ellipse corresponds to a suppressive stimulus direction: stimuli with a significant component along this axis are less likely to elicit spikes. The stimulus component along the major axis of the ellipse has no influence on spiking.

Here we develop a white noise methodology for characterizing a neuron with gain control. We show that a set of suppressive kernels may be recovered by finding the eigenvectors of the spike-triggered covariance matrix associated with *smallest* variance. We apply the technique to electrophysiological data obtained from ganglion cells in salamander and macaque retina, and recover a set of axes that are shown to reduce responses in the neuron. Moreover, when we fit a gain control model to the data using a maximum likelihood procedure within this subspace, the model accounts for changes in the STA as a function of contrast.

# 1    Characterizing suppressive axes

As in all white noise approaches, we assume that stimuli correspond to vectors, $\vec{s}$, in some finite-dimensional space (e.g., a neighborhood of pixels or an interval of time samples). We assume a gain control model in which the probability of a stimulus eliciting a spike grows monotonically with the halfwave-rectified projection onto an excitatory linear kernel, $\lfloor \vec{k}_0 \cdot \vec{s} \rfloor$, and is suppressively modulated by the fullwave-rectified projection onto a set of linear kernels, $|\vec{k}_n \cdot \vec{s}|$.

First, we recover the excitatory kernel, $\vec{k}_0$. This is achieved by presenting spherically symmetric input stimuli (e.g., Gaussian white noise) to the neuron and computing the STA (Fig. 1a). STA correctly recovers the excitatory kernel, under the assumption that each of the gain control kernels are orthogonal (or equal) to the excitatory kernel. The proof is essentially the same as that given for recovering the kernel of a linear model followed by a monotonic nonlinearity [3]. In particular, any stimulus can be decomposed into a component in the direction of the excitatory kernel, and a component in a perpendicular direction. This can be paired with another stimulus that is identical, except that its component in the perpendicular direction is negated. The two stimuli are equally likely to occur in a spherically Gaussian stimulus set (since they are equidistant from the origin), and they are equally likely to elicit a spike (since their excitatory components are equal, and their rectified perpendicular components are equal). Their vector average lies in the direction of the excitatory kernel. Thus, the STA (which is an average over all such stimuli, or all such stimulus pairs) must also lie in that direction. In a subsequent section we explain how to

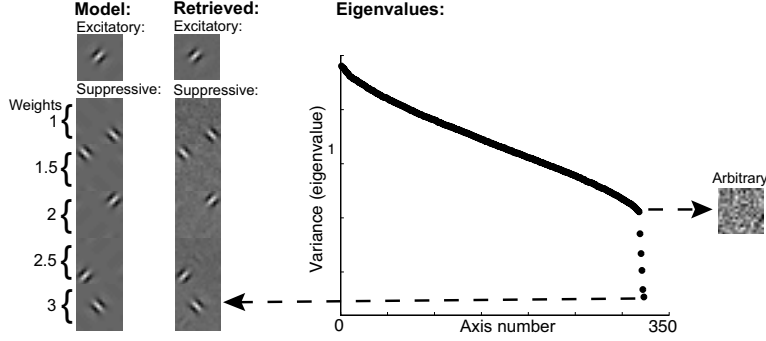

**Figure 2:** Estimation of kernels from a simulated model (equation 2). Left: Model kernels. Right: Sorted eigenvalues of covariance matrix of stimuli eliciting spikes (STC). Five eigenvalues fall significantly below the others. Middle: STA (excitatory kernel) and eigenvectors (suppressive kernels) associated with the lowest eigenvalues.

recover the excitatory kernel when it is not orthogonal to the suppressive kernels.

Next, we recover the suppressive subspace, assuming the excitatory kernel is known. Consider the stimuli lying on a plane perpendicular to this kernel. These stimuli all elicit the same response in the excitatory kernel, but they may produce different amounts of suppression. Figure 1b illustrates the behavior in a three-dimensional stimulus space, in which one axis is assumed to be suppressive. The distribution of raw stimuli on the plane is spherically symmetric about the origin. But the distribution of stimuli eliciting a spike is narrower along the suppressive direction: these stimuli have a component along the suppressive axis and are therefore less likely to elicit a spike. This behavior is easily generalized from this plane to the entire stimulus space. If we assume that the suppressive axes are fixed, then we expect to see reductions in variance in the same directions for any level of numerator excitation.

Given this behavior of the spike-triggered stimulus ensemble, we can recover the suppressive subspace using principal component analysis. We construct the sample covariance matrix of the stimuli eliciting a spike:

$$C = \frac{1}{N_s - 1} \sum_{\vec{s}_{\text{spike}}} \vec{s}\,\vec{s}^T, \tag{1}$$

where $N_s$ is the number of spikes. To ensure the estimated suppressive subspace is orthogonal to the estimated $\vec{k}_0$ (as in Figure 1b), the stimuli $\vec{s}_{\text{spike}}$ are first projected onto the subspace perpendicular to the estimated $\vec{k}_0$. The principal axes (eigenvectors) of $C$ that are associated with small variance (eigenvalues) correspond to directions in which the response of the neuron is modulated suppressively.

We illustrate the technique on simulated data for a neuron with a spatio-temporal receptive field. The kernels are a set of orthogonal bandpass filters. The stimulus vectors $\vec{s}$ of this input sequence are defined over a 18-sample spatial region and a 18-sample time window (i.e., a 324-dimensional space). Spikes are generated using a Poisson process with mean rate determined by a specific form of gain control [14]:

$$\mathcal{P}\left(spike|\vec{s}\right) = \frac{\lfloor \vec{k}_0 \cdot \vec{s} \rfloor^2}{\sum_n w_n |\vec{k}_n \cdot \vec{s}|^2 + \sigma^2}. \tag{2}$$

The goal of simulation is to recover excitatory kernel $\vec{k}_0$, the suppressive subspace spanned by $\vec{k}_n$, weights $w_n$, and constant $\sigma$.

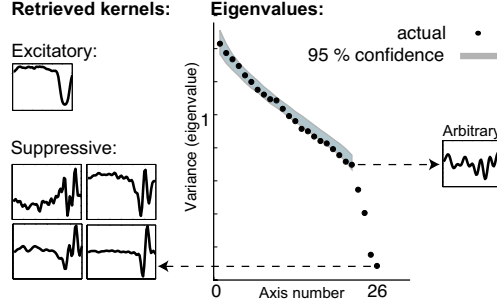

**Figure 3:** Left: Retrieved kernels from STA and STC analysis of ganglion cell data from a salamander retina (cell 1999-11-12-B6A). Right: sorted eigenvalues of the spike-triggered covariance matrix, with corresponding eigenvectors. Low eigenvalues correspond to suppressive directions, while other eigenvalues correspond to arbitrary (ignored) directions. Raw stimulus ensemble was sphered (whitened) prior to analysis and low-variance axes underrepresented in stimulus set were discarded.

Figure 2 shows the original and estimated kernels for a model simulation with 600K input samples and 36.98K spikes. First, we note that STA recovers an accurate estimate of the excitatory kernel. Next, consider the sorted eigenvalues of $C$, as plotted in Figure 2. The majority of the eigenvalues descend gradually (the covariance matrix of the white noise source should have constant eigenvalues, but remember that those in Figure 2 are computed from a finite set of samples). The last five eigenvalues are significantly below the values one would obtain with randomly selected stimulus subsets. The eigenvectors associated with these lowest eigenvalues span approximately the same subspace as the suppressive kernels. Note that some eigenvectors correspond to mixtures of the original suppressive kernels, due to non-uniqueness of the eigenvector decomposition. In contrast, eigenvectors corresponding to eigenvalues in the gradually-descending region appear arbitrary in their structure.

Finally, we can recover the scalar parameters of this specific model ($w_n$ and $\sigma$) by selecting them to maximize the likelihood of the spike data according to equation (2). Note that a direct maximum likelihood solution on the raw data would have been impractical due to the high dimensionality of the stimulus space.

## 2  Suppressive Axes in Retinal Ganglion Cells

Retinal ganglion cells exhibit rapid [8, 15] as well as slow [9, 16, 17] gain control. We now demonstrate that we can recover a rapid gain control signal by applying the method to data from salamander retina [9]. The input sequence consists of 80K time samples of full-field 33Hz flickering binary white noise (contrast 8.5%). The stimulus vectors $\vec{s}$ of this sequence are defined over a 60-segment time window. Since stimuli are finite in number and binary, they are not spherically distributed. To correct for this, we discard low-variance axes and whiten the stimuli within the remaining axes.

Figure 3 depicts the kernels estimated from the 623 stimulus vectors eliciting spikes. Similar to the model simulation, the eigenvalues gradually fall off, but four of the eigenvalues appear to drop significantly below the rest. To make this more concrete, we test the hypothesis that the majority of the eigenvalues are consistent with those of randomly selected stimulus vectors, but that the last 4 eigenvalues fall significantly below this range. Specifically, we perform a Monte Carlo simulation, drawing (with replacement) random subsets of 623 stimuli from the full set of raw stimuli. We also randomly select 4 (orthogonal)

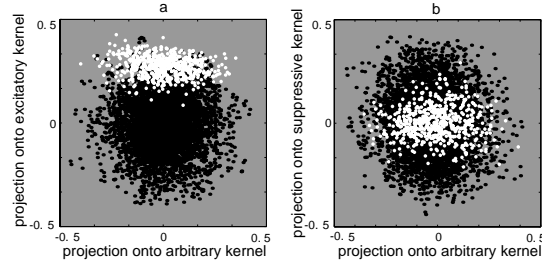

**Figure 4:** Scatter plots from salamander ganglion cell data (cell 1999-11-12-B6A). Black points indicate the raw stimulus set. White points indicate stimuli eliciting a spike. **a**, Projection of stimuli onto estimated excitatory kernel vs. arbitrary kernel. **b**, Projection of stimuli onto an estimated suppressive kernel vs. arbitrary kernel.

axes, representing a suppressive subspace, and project this subspace out of the set of randomly chosen stimuli. We then compute the eigenvalues of the sample covariance matrix of these stimuli. We repeat this 1000 times, and estimate a 95 percent confidence interval for each of the eigenvalues. The figure shows that the first eigenvalues lie within the confidence interval. In practice, we repeat this process in a nested fashion, assuming initially no directions are significantly suppressive, then one direction, and so on up to four directions.

These low eigenvalues correspond to eigenvectors that are concentrated in recent time (as is the estimated excitatory kernel). The remaining eigenvectors appear to be arbitrary, spanning the full temporal window. We emphasize that these kernels should not be interpreted to correspond to receptive fields of individual neurons underlying the suppressive signal, but merely provide an orthogonal basis for a suppressive subspace.

We can now verify that the recovered STA axis is in fact excitatory, and the kernels corresponding to the lowest eigenvalues are suppressive. Figure 4a shows a scatter plot of the stimuli projected onto the excitatory axis vs. an arbitrary axis. Spikes are seen to occur only when the component along the excitatory axis is high, as expected. Figure 4b is a scatter plot of the stimuli projected onto one of the suppressive axes vs. an arbitrary (ignored) axis. The spiking stimuli lie within an ellipse, with the minor axis corresponding to the suppressive kernel. This is exactly what we would expect in a suppressive gain control system (see Figure 1b).

Figure 5 illustrates recovery of a two-dimensional suppressive subspace for a macaque retinal ganglion cell. The subspace was computed from the 36.43K stimulus vectors eliciting spikes out of a total of 284.74K vectors. The data are qualitatively similar to those of the salamander cell, although both the strength of suppression and specific shapes of the scatter plots differs. In addition to suppression, the method recovers facilitation (i.e., high-variance axes) in some cells (not shown here).

## 3 Correcting for Bias in Kernel Estimates

The kernels in the previous section were all recovered from stimuli of a single contrast. However, when the STA is computed in a ganglion cell for low and high contrast stimuli, the low-contrast kernel shows a slower time course [9] (figure 7,a). This would appear inconsistent with the method we describe, in which the STA is meant to provide an estimate of a single excitatory kernel. This behavior can be explained by assuming a model of the form given in equation 2, and in addition dropping the constraint that the gain control kernels are orthogonal (or identical) to the excitatory kernel.

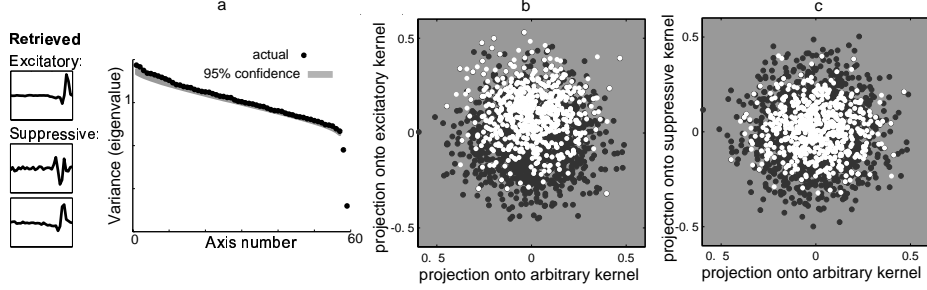

**Figure 5: a**, Sorted eigenvalues of stimuli eliciting spikes from a macaque retina (cell 2001-09-29-E6A). **b-c**, Scatter plots of stimuli projected onto recovered axes.

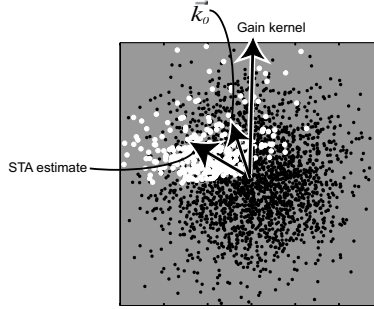

**Figure 6:** Demonstration of estimator bias. When a gain control kernel is not orthogonal to the excitatory kernel, the responses to one side of the excitatory kernel are suppressed more than those on the other side. The resulting STA estimate is thus biased away from the true excitatory kernel, $\vec{k}_0$.

First we show that when the orthogonality constraint is dropped, the STA estimate of the excitatory kernel is biased by the gain control signal. Consider a situation in which a suppressive kernel contains a component in the direction of the excitatory kernel, $\vec{k}_0$. We write $\vec{k}_n = \alpha \vec{k}_0 + \vec{k}'_n$, where $\vec{k}'_n$ is perpendicular to the excitatory kernel. Then, for example, a stimulus $\vec{s} = \vec{k}_0 + \beta \vec{k}'_n$, with $\beta > 0$, produces a suppressive component along $\vec{k}_n$ equal to $\alpha ||\vec{k}_0||^2 + \beta ||\vec{k}'_n||^2$, but the corresponding paired stimulus vector $\vec{s} = \vec{k}_0 - \beta \vec{k}'_n$ produces a suppressive component of $\alpha ||\vec{k}_0||^2 - \beta ||\vec{k}'_n||^2$. Thus, the two stimuli are equally likely to occur but not equally likely to elicit a spike. As a result, the STA will be biased in the direction $-\vec{k}'_n$. Figure 6 illustrates an example in which a non-orthogonal suppressive axis biases the estimate of the STA.

Now consider the model in equation 2 in the presence of a non-orthogonal suppressive subspace. Note that the bias is stronger for larger amplitude stimuli because the constant term $\sigma^2$ dominates the gain control signal for weak stimuli. Indeed, we have previously hypothesized that changes in receptive field tuning can arise from divisive gain control models that include an additive constant [14].

Even when the STA estimate is biased by the gain control signal, we can still obtain an (asymptotically) unbiased estimate of the excitatory kernel. Specifically, the true excitatory kernel lies within the subspace spanned by the estimated (biased) excitatory and suppressive kernels. So, assuming a particular gain control model, we can again maximize the likelihood of the data, but now allowing both the excitatory and suppressive kernels to move within the subspace spanned by the initial estimated kernels. The resulting suppres-

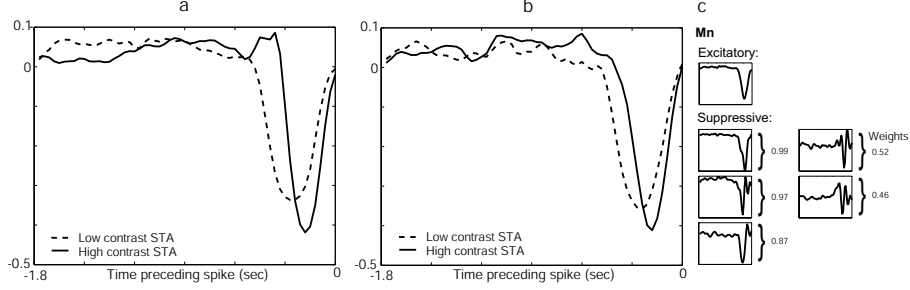

**Figure 7:** STA kernels estimated from low (8.5%) and high (34%) contrast salamander retinal ganglion cell data (cell 1999-11-12-B6A). Kernels are normalized to unit energy. **a**, STA kernels derived from ganglion cell spikes. **b**, STA kernels derived from simulated spikes using ML-estimated model. **c**, Kernels and corresponding weights of ML-estimated model.

sive kernels need not be orthogonal to the excitatory kernel.

We maximize the likelihood of the full two-contrast data set using a model that is a generalization of that given by equation (2):

$$\mathcal{P}\left(spike|\vec{s}\right) = \frac{\lfloor \vec{k}_0 \cdot \vec{s} \rfloor^p}{(\sum_n w_n |\vec{k}_n \cdot \vec{s}|^2)^{p/2} + \sigma^p} \tag{3}$$

The exponent $p$ is incorporated to allow for more realistic contrast-response functions. The excitatory axis is initially set to the STA and the suppressive axes are set to the low-eigenvalue eigenvectors of the STC, along with the STA (e.g., to allow for self-suppression). The recovered axes and weights are shown in Figure 7b, and remaining model parameters are: $p = 7.18$, $\sigma = 0.4126$. Whereas the axes recovered from the STA/STC analysis are orthogonal, the axes determined during the maximum likelihood stage need not be (and in the data example are not) orthogonal. Figure 7b also demonstrates that the fitted model accounts for the change in STA observed at different contrast levels. Specifically, we simulate responses of the model (equation (3) with Poisson spike generation) on each of the two contrast stimulus sets, and then compute the STA based on these simulated spike trains. Although it is based on a single fixed excitatory kernel, the model exhibits a change in STA shape as a function of contrast very much like the salamander neuron.

## 4   Discussion

We have described a spike-triggered covariance method for characterizing a neuron with gain control, and demonstrated the plausibility of the technique through simulation and analysis of neural data. The suppressive axes recovered from retinal ganglion cell data appear to be significant because: (1) As in the model simulation, a small number of eigenvalues are significantly below the rest; (2) The eigenvectors associated with these axes are concentrated in a temporal region immediately preceding the spike, unlike the remaining axes; (3) Projection of the multi-dimensional stimulus vectors onto these axes reveal reductions of spike probability; (4) The full model, with parameters recovered through maximum likelihood, explains changes in STA as a function of contrast.

Models of retinal processing often incorporate gain control [e.g., 8, 10, 15, 17, 18]. We have shown for the first time how one can use white noise analysis to recover a gain control subspace. The kernels defining this subspace correspond to relatively short timescales. Thus, it is interesting to compare the recovered subspace to models of rapid gain control. In particular, Victor [15] proposed a retinal gain model in which the gain signal consists

of time-delayed copies of the excitatory kernel. In fact, for the cell shown in Figure 3, the recovered suppressive subspace lies within the space spanned by shifted copies of the excitatory kernel. The fact that we do not see evidence for slow gain control in the analysis might indicate that these signals do not lie within a low-dimensional stimulus subspace. In addition, the analysis is not capable of distinguishing between physiological mechanisms that could underlie gain control behaviors. Potential candidates may include internal bio-chemical adjustments, non-Poisson spike generation mechanisms, synaptic depression, and shunting inhibition due to other neurons.

This technique should be applicable to a far wider range of neural data than has been shown here. Future work will incorporate analysis of data gathered using stimuli that vary in both time and space (as in the simulated example of Figure 2). We are also exploring applicability of the technique to other visual areas.

**Acknowledgments**    We thank Liam Paninski and Jonathan Pillow for helpful discussions and comments, and Divya Chander for data collection.

# References

[1]  E deBoer and P Kuyper. Triggered correlation. In *IEEE Transact. Biomed. Eng.*, volume 15, pages 169–179, 1968.

[2]  J P Jones and L A Palmer. The two-dimensional spatial structure of simple receptive fields in the cat striate cortex. *J Neurophysiology*, 58:1187–11211, 1987.

[3]  E J Chichilnisky. A simple white noise analysis of neuronal light responses. *Network: Computation in Neural Systems*, 12(2):199–213, 2001.

[4]  D L Ringach, G Sapiro, and R Shapley. A subspace reverse-correlation technique for the study of visual neurons. *Vision Research*, 37:2455–2464, 1997.

[5]  R de Ruyter van Steveninck and W Bialek. Coding and information transfer in short spike sequences. In *Proc.Soc. Lond. B. Biol. Sci.*, volume 234, pages 379–414, 1988.

[6]  B A Y Arcas, A L Fairhall, and W Bialek. What can a single neuron compute? In *Advances in Neural Information Processing Systems*, volume 13, pages 75–81, 2000.

[7]  M Carandini, D J Heeger, and J A Movshon. Linearity and normalization in simple cells of the macaque primary visual cortex. *Journal of Neuroscience*, 17:8621–8644, 1997.

[8]  R M Shapley and J D Victor. The effect of contrast on the transfer properties of cat retinal ganglion cells. *J. Physiol. (Lond)*, 285:275–298, 1978.

[9]  D Chander and E J Chichilnisky. Adaptation to temporal contrast in primate and salamander retina. *J Neurosci*, 21(24):9904–9916, 2001.

[10]  R Shapley and C Enroth-Cugell. Visual adaptation and retinal gain control. *Progress in Retinal Research*, 3:263–346, 1984.

[11]  R F Lyon. Automatic gain control in cochlear mechanics. In P Dallos et al., editor, *The Mechanics and Biophysics of Hearing*, pages 395–420. Springer-Verlag, 1990.

[12]  W S Geisler and D G Albrecht. Cortical neurons: Isolation of contrast gain control. *Vision Research*, 8:1409–1410, 1992.

[13]  D J Heeger. Normalization of cell responses in cat striate cortex. *Vis. Neuro.*, 9:181–198, 1992.

[14]  O Schwartz and E P Simoncelli. Natural signal statistics and sensory gain control. *Nature Neuroscience*, 4(8):819–825, August 2001.

[15]  J D Victor. The dynamics of the cat retinal X cell centre. *J. Physiol.*, 386:219–246, 1987.

[16]  S M Smirnakis, M J Berry, David K Warland, W Bialek, and M Meister. Adaptation of retinal processing to image contrast and spatial scale. *Nature*, 386:69–73, March 1997.

[17]  K J Kim and F Rieke. Temporal contrast adaptation in the input and output signals of salamander retinal ganglion cells. *J. Neurosci.*, 21(1):287–299, 2001.

[18]  M Meister and M J Berry. The neural code of the retina. *Neuron*, 22:435–450, 1999.
